# Rademacher Complexity Bounds
# for Non-I.I.D. Processes

**Mehryar Mohri**
Courant Institute of Mathematical Sciences
and Google Research
251 Mercer Street
New York, NY 10012
mohri@cims.nyu.edu

**Afshin Rostamizadeh**
Department of Computer Science
Courant Institute of Mathematical Sciences
251 Mercer Street
New York, NY 10012
rostami@cs.nyu.edu

## Abstract

This paper presents the first Rademacher complexity-based error bounds for non-i.i.d. settings, a generalization of similar existing bounds derived for the i.i.d. case. Our bounds hold in the scenario of dependent samples generated by a stationary $\beta$-mixing process, which is commonly adopted in many previous studies of non-i.i.d. settings. They benefit from the crucial advantages of Rademacher complexity over other measures of the complexity of hypothesis classes. In particular, they are data-dependent and measure the complexity of a class of hypotheses based on the training sample. The empirical Rademacher complexity can be estimated from such finite samples and lead to tighter generalization bounds. We also present the first margin bounds for kernel-based classification in this non-i.i.d. setting and briefly study their convergence.

## 1 Introduction

Most learning theory models such as the standard PAC learning framework [13] are based on the assumption that sample points are independently and identically distributed (i.i.d.). The design of most learning algorithms also relies on this key assumption. In practice, however, the i.i.d. assumption often does not hold. Sample points have some temporal dependence that can affect the learning process. This dependence may appear more clearly in times series prediction or when the samples are drawn from a Markov chain, but various degrees of time-dependence can also affect other learning problems.

A natural scenario for the analysis of non-i.i.d. processes in machine learning is that of observations drawn from a stationary mixing sequence, a standard assumption adopted in most previous studies, which implies a dependence between observations that diminishes with time [7,9,10,14,15]. The pioneering work of Yu [15] led to VC-dimension bounds for stationary $\beta$-mixing sequences. Similarly, Meir [9] gave bounds based on covering numbers for time series prediction [9]. Vidyasagar [14] studied the extension of PAC learning algorithms to these non-i.i.d. scenarios and proved that under some sub-additivity conditions, a PAC learning algorithm continues to be PAC for these settings. Lozano et al. studied the convergence and consistency of regularized boosting under the same assumptions [7]. Generalization bounds have also been derived for stable algorithms with weakly dependent observations [10]. The consistency of learning under the more general scenario of $\alpha$-mixing with non-stationary sequences has also been studied by Irle [3] and Steinwart et al. [12].

This paper gives data-dependent generalization bounds for stationary $\beta$-mixing sequences. Our bounds are based on the notion of Rademacher complexity. They extend to the non-i.i.d. case the Rademacher complexity bounds derived in the i.i.d. setting [2, 4, 5]. To the best of our knowledge, these are the first Rademacher complexity bounds derived for non-i.i.d. processes. Our proofs make

use of the so-called *independent block technique* due to Yu [15] and Bernstein and extend the applicability of the notion of Rademacher complexity to non-i.i.d. cases.

Our generalization bounds benefit from all the advantageous properties of Rademacher complexity as in the i.i.d. case. In particular, since the Rademacher complexity can be bounded in terms of other complexity measures such as covering numbers and VC-dimension [1], it allows us to derive generalization bounds in terms of these other complexity measures, and in fact improve on existing bounds in terms of these other measures, e.g., VC-dimension. But, perhaps the most crucial advantage of bounds based on the empirical Rademacher complexity is that they are data-dependent: they measure the complexity of a class of hypotheses based on the training sample and thus better capture the properties of the distribution that has generated the data. The empirical Rademacher complexity can be estimated from finite samples and lead to tighter bounds. Furthermore, the Rademacher complexity of large hypothesis sets such as kernel-based hypotheses, decision trees, convex neural networks, can sometimes be bounded in some specific ways [2]. For example, the Rademacher complexity of kernel-based hypotheses can be bounded in terms of the trace of the kernel matrix.

In Section 2, we present the essential notion of a mixing process for the discussion of learning in non-i.i.d. cases and define the learning scenario. Section 3 introduces the idea of independent blocks and proves a bound on the expected deviation of the error from its empirical estimate. In Section 4, we present our main Rademacher generalization bounds and discuss their properties.

## 2 Preliminaries

This section introduces the concepts needed to define the non-i.i.d. scenario we will consider, which coincides with the assumptions made in previous studies [7, 9, 10, 14, 15].

### 2.1 Non-I.I.D. Distributions

The non-i.i.d. scenario we will consider is based on *stationary $\beta$-mixing processes*.

**Definition 1** (Stationarity)**.** *A sequence of random variables* $\mathbf{Z} = \{Z_t\}_{t=-\infty}^{\infty}$ *is said to be* stationary *if for any $t$ and non-negative integers $m$ and $k$, the random vectors $(Z_t, \ldots, Z_{t+m})$ and $(Z_{t+k}, \ldots, Z_{t+m+k})$ have the same distribution.*

Thus, the index $t$ or time, does not affect the distribution of a variable $Z_t$ in a stationary sequence (note that this does not imply independence).

**Definition 2** ($\beta$-mixing)**.** *Let $\mathbf{Z} = \{Z_t\}_{t=-\infty}^{\infty}$ be a stationary sequence of random variables. For any $i, j \in \mathbb{Z} \cup \{-\infty, +\infty\}$, let $\sigma_i^j$ denote the $\sigma$-algebra generated by the random variables $Z_k$, $i \leq k \leq j$. Then, for any positive integer $k$, the $\beta$-mixing coefficient of the stochastic process $\mathbf{Z}$ is defined as*

$$\beta(k) = \sup_{n} \; \mathbf{E}_{B \in \sigma_{-\infty}^{n}} \left[ \sup_{A \in \sigma_{n+k}^{\infty}} \left| \Pr[A \mid B] - \Pr[A] \right| \right]. \tag{1}$$

*$\mathbf{Z}$ is said to be $\beta$-mixing if $\beta(k) \to 0$. It is said to be* algebraically $\beta$-mixing *if there exist real numbers $\beta_0 > 0$ and $r > 0$ such that $\beta(k) \leq \beta_0/k^r$ for all $k$, and* exponentially mixing *if there exist real numbers $\beta_0$ and $\beta_1$ such that $\beta(k) \leq \beta_0 \exp(-\beta_1 k^r)$ for all $k$.*

Thus, a sequence of random variables is mixing when the dependence of an event on those occurring $k$ units of time in the past weakens as a function of $k$.

### 2.2 Rademacher Complexity

Our generalization bounds will be based on the following measure of the complexity of a class of functions.

**Definition 3** (Rademacher Complexity)**.** *Given a sample $S \in X^m$, the empirical Rademacher complexity of a set of real-valued functions $H$ defined over a set $X$ is defined as follows:*

$$\widehat{\mathfrak{R}}_S(H) = \frac{2}{m} \mathbf{E}_{\sigma} \left[ \sup_{h \in H} \left| \sum_{i=1}^{m} \sigma_i h(x_i) \right| \; \middle| \; S = (x_1, \ldots, x_m) \right]. \tag{2}$$

*The expectation is taken over $\sigma = (\sigma_1, \ldots, \sigma_n)$ where $\sigma_i s$ are independent uniform random variables taking values in $\{-1, +1\}$ called Rademacher random variables. The Rademacher complexity of a hypothesis set $H$ is defined as the expectation of $\widehat{\mathfrak{R}}_S(H)$ over all samples of size $m$:*

$$\mathfrak{R}_m(H) = \mathop{\mathbf{E}}_S\left[\widehat{\mathfrak{R}}_S(H)\big|\,|S| = m\right]. \tag{3}$$

The definition of the Rademacher complexity depends on the distribution according to which samples $S$ of size $m$ are drawn, which in general is a dependent $\beta$-mixing distribution $D$. In the rare instances where a different distribution $\widetilde{D}$ is considered, typically for an i.i.d. setting, we explicitly indicate that distribution as a superscript: $\mathfrak{R}_m^{\widetilde{D}}(H)$.

The Rademacher complexity measures the ability of a class of functions to fit noise. The empirical Rademacher complexity has the added advantage that it is data-dependent and can be measured from finite samples. This can lead to tighter bounds than those based on other measures of complexity such as the VC-dimension [2, 4, 5].

We will denote by $\widehat{R}_S(h)$ the empirical average of a hypothesis $h \colon X \to \mathbb{R}$ and by $R(h)$ its expectation over a sample $S$ drawn according to a stationary $\beta$-mixing distribution:

$$\widehat{R}_S(h) = \frac{1}{m}\sum_{i=1}^{m} h(z_i) \qquad R(h) = \mathop{\mathbf{E}}_S[\widehat{R}_S(h)]. \tag{4}$$

The following proposition shows that this expectation is independent of the size of the sample $S$, as in the i.i.d. case.

**Proposition 1.** *For any sample $S$ of size $m$ drawn from a stationary distribution $\mathcal{D}$, the following holds: $\mathbf{E}_{S \sim \mathcal{D}^m}[\widehat{R}_S(h)] = \mathbf{E}_{z \sim \mathcal{D}}[h(z)]$.*

*Proof.* Let $S = (x_1, \ldots, x_m)$. By stationarity, $\mathbf{E}_{z_i \sim \mathcal{D}}[h(z_i)] = \mathbf{E}_{z_j \sim \mathcal{D}}[h(z_j)]$ for all $1 \le i, j \le m$, thus, we can write:

$$\mathop{\mathbf{E}}_S[\widehat{R}_S(h)] = \frac{1}{m}\sum_{i=1}^{m} \mathop{\mathbf{E}}_S[h(z_i)] = \frac{1}{m}\sum_{i=1}^{m} \mathop{\mathbf{E}}_{z_i}[h(z_i)] = \mathop{\mathbf{E}}_z[h(z)]. \qquad \square$$

## 3 Proof Components

Our proof makes use of McDiarmid's inequality [8] to show that the empirical average closely estimates its expectation. To derive a Rademacher generalization bound, we apply McDiarmid's inequality to the following random variable, which is the quantity we wish to bound:

$$\Phi(S) = \sup_{h \in H} R(h) - \widehat{R}_S(h). \tag{5}$$

McDiarmid's inequality bounds the deviation of $\Phi$ from its mean, thus, we must also bound the expectation $\mathbf{E}[\Phi]$. However, we immediately face two obstacles: both McDiarmid's inequality and the standard bound on $\mathbf{E}[\Phi]$ hold only for samples drawn in an i.i.d. fashion. The main idea behind our proof is to analyze the non-i.i.d. setting and transfer it to a close independent setting. The following sections will describe in detail our solution to these problems.

### 3.1 Independent Blocks

We derive Rademacher generalization bounds for the case where training and test points are drawn from a stationary $\beta$-mixing sequence. As in previous non-i.i.d. analyses [7, 9, 10, 15], we use a technique transferring the original problem based on dependent points to one based on a sequence of *independent blocks*. The method consists of first splitting a sequence $S$ into two subsequences $S_0$ and $S_1$, each made of $\mu$ blocks of $a$ consecutive points. Given a sequence $S = (z_1, \ldots, z_m)$ with $m = 2a\mu$, $S_0$ and $S_1$ are defined as follows:

$$S_0 = (Z_1, Z_2, \ldots, Z_\mu), \qquad \text{where } Z_i = (z_{(2i-1)+1}, \ldots, z_{(2i-1)+a}), \tag{6}$$

$$S_1 = (Z_1^{(1)}, Z_2^{(1)}, \ldots, Z_\mu^{(1)}), \qquad \text{where } Z_i^{(1)} = (z_{2i+1}, \ldots, z_{2i+a}). \tag{7}$$

Instead of the original sequence of odd blocks $S_0$, we will be working with a sequence $\widetilde{S}_0$ of *independent* blocks of equal size $a$ to which standard i.i.d. techniques can be applied: $\widetilde{S}_0 = (\widetilde{Z}_1, \widetilde{Z}_2, \ldots, \widetilde{Z}_\mu)$ with mutually independent $\widetilde{Z}_k$s, but, the points within each block $\widetilde{Z}_k$ follow the same distribution as in $Z_k$. As stated by the following result of Yu [15][Corollary 2.7], for a sufficiently large spacing $a$ between blocks and a sufficiently fast mixing distribution, the expectation of a bounded measurable function $h$ is essentially unchanged if we work with $\widetilde{S}_0$ instead of $S_0$.

**Corollary 1** ([15]). *Let $h$ be a measurable function bounded by $M \geq 0$ defined over the blocks $Z_k$, then the following holds:*

$$| \underset{S_0}{\mathbf{E}}[h] - \underset{\widetilde{S}_0}{\mathbf{E}}[h]| \leq (\mu - 1)M\beta(a), \tag{8}$$

*where $\mathbf{E}_{S_0}$ denotes the expectation with respect to $S_0$, $\mathbf{E}_{\widetilde{S}_0}$ the expectation with respect to the $\widetilde{S}_0$.*

We denote by $\widetilde{D}$ the distribution corresponding to the independent blocks $\widetilde{Z}_k$. Also, to work with block sequences, we extend some of our definitions: we define the extension $h_a \colon Z^a \to \mathbb{R}$ of any hypothesis $h \in H$ to a block-hypothesis by $h_a(B) = \frac{1}{a}\sum_{i=1}^{a} h(Z_i)$ for any block $B = (z_1, \ldots, z_a) \in Z^a$, and define $H_a$ as the set of all block-based hypotheses $h_a$ generated from $h \in H$.

It will also be useful to define the subsequence $S_\mu$, which consists of $\mu$ singleton points separated by a gap of $2a - 1$ points. This can be thought of as the sequence constructed from $S_0$, or $S_1$, by selecting only the $j$th point from each block, for any fixed $j \in \{1, \ldots, a\}$.

## 3.2 Concentration Inequality

McDiarmid's inequality requires the sample to be i.i.d. Thus, we first show that $\Pr[\Phi(S)]$ can be bounded in terms of independent blocks and then apply McDiarmid's inequality to the independent blocks.

**Lemma 1.** *Let $H$ be a set of hypotheses bounded by $M$. Let $S$ denote a sample, of size $m$, drawn according to a stationary $\beta$-mixing distribution and let $\widetilde{S}_0$ denote a sequence of independent blocks. Then, for all $a, \mu, \epsilon > 0$ with $2\mu a = m$ and $\epsilon > \mathbf{E}_{\widetilde{S}_0}[\Phi(\widetilde{S}_0)]$, the following bound holds:*

$$\underset{S}{\Pr}[\Phi(S) > \epsilon] \leq 2 \underset{\widetilde{S}_0}{\Pr}[\Phi(\widetilde{S}_0) - \underset{\widetilde{S}_0}{\mathbf{E}}[\Phi(\widetilde{S}_0)] > \epsilon'] + 2(\mu - 1)\beta(a),$$

*where $\epsilon' = \epsilon - \mathbf{E}_{\widetilde{S}_0}[\Phi(\widetilde{S}_0)]$.*

*Proof.* We first rewrite the left-hand side probability in terms of even and odd blocks and then apply Corollary 1 as follows:

$$\underset{S}{\Pr}[\Phi(S) > \epsilon] = \underset{S}{\Pr}[\sup_h(R(h) - \widehat{R}_S(h)) > \epsilon]$$

$$= \underset{S}{\Pr}\left[\sup_h\left(\frac{R(h) - \widehat{R}_{S_0}(h)}{2} + \frac{R(h) - \widehat{R}_{S_1}(h)}{2}\right) > \epsilon\right] \qquad \text{(def. of } \widehat{R}_S(h)\text{)}$$

$$\leq \underset{S}{\Pr}\left[\frac{1}{2}\left(\sup_h(R(h) - \widehat{R}_{S_0}(h)) + \sup_h(R(h) - \widehat{R}_{S_1}(h))\right) > \epsilon\right] \quad \text{(convexity of sup)}$$

$$= \underset{S}{\Pr}[\Phi(S_0) + \Phi(S_1) > 2\epsilon] \qquad \text{(def. of } \Phi\text{)}$$

$$\leq \underset{S_0}{\Pr}[\Phi(S_0) > \epsilon] + \underset{S_1}{\Pr}[\Phi(S_1) > \epsilon] \qquad \text{(union bound)}$$

$$= 2 \underset{S_0}{\Pr}[\Phi(S_0) > \epsilon] \qquad \text{(stationarity)}$$

$$= 2 \underset{S_0}{\Pr}[\Phi(S_0) - \underset{\widetilde{S}_0}{\mathbf{E}}[\Phi(\widetilde{S}_0)] > \epsilon']. \qquad \text{(def. of } \epsilon'\text{)}$$

The second inequality holds by the union bound and the fact that $\Phi(S_0)$ or $\Phi(S_1)$ must surpass $\epsilon$ for their sum to surpass $2\epsilon$. To complete the proof, we apply Corollary 1 to the expectation of the indicator variable of the event $\{\Phi(S_0) - \mathbf{E}_{\widetilde{S}_0}[\Phi(\widetilde{S}_0)] > \epsilon'\}$, which yields

$$2 \underset{S_0}{\Pr}[\Phi(S_0) - \underset{\widetilde{S}_0}{\mathbf{E}}[\Phi(\widetilde{S}_0)] > \epsilon'] \leq 2 \underset{\widetilde{S}_0}{\Pr}[\Phi(\widetilde{S}_0) - \underset{\widetilde{S}_0}{\mathbf{E}}[\Phi(\widetilde{S}_0)] > \epsilon'] + 2(\mu - 1)\beta(a). \qquad \square$$

We can now apply McDiarmid's inequality to the independent blocks of Lemma 1.

**Proposition 2.** *For the same assumptions as in Lemma 1, the following bound holds for all $\epsilon >$*
$\mathbf{E}_{\widetilde{S}_0}[\Phi(\widetilde{S}_0)]$:

$$\Pr_S[\Phi(S) > \epsilon] \leq 2\exp\left(\frac{-2\mu\epsilon'^2}{M^2}\right) + 2(\mu - 1)\beta(a),$$

*where $\epsilon' = \epsilon - \mathbf{E}_{\widetilde{S}_0}[\Phi(\widetilde{S}_0)]$.*

*Proof.* To apply McDiarmid's inequality, we view each block as an i.i.d. *point* with respect to $h_a$. $\Phi(\widetilde{S}_0)$ can be written in terms of $h_a$ as: $\Phi(\widetilde{S}_0) = R(h_a) - \widehat{R}_{\widetilde{S}_0}(h_a) = R(h_a) - \frac{1}{\mu}\sum_{k=1}^{\mu} h_a(\widetilde{Z}_k)$. Thus, changing a block $\widetilde{Z}_k$ of the sample $\widetilde{S}_0$ can change $\Phi(\widetilde{S}_0)$ by at most $\frac{1}{\mu}|h(\widetilde{Z}_k)| \leq M/\mu$. By McDiarmid's inequality, the following holds for any $\epsilon > 2(\mu - 1)M\beta(a)$:

$$\Pr_{\widetilde{S}_0}[\Phi(\widetilde{S}_0) - \mathbf{E}_{\widetilde{S}_0}[\Phi(\widetilde{S}_0)] > \epsilon'] \leq \exp\left(\frac{-2\epsilon'^2}{\sum_{i=1}^{\mu}(M/\mu)^2}\right) = \exp\left(\frac{-2\mu\epsilon'^2}{M^2}\right).$$

Plugging in the right-hand side in the statement of Lemma 1 proves the proposition. $\qquad\square$

### 3.3  Bound on the Expectation

Here, we give a bound on $\mathbf{E}_{\widetilde{S}_0}[\Phi(S_0)]$ based on the Rademacher complexity, as in the i.i.d. case [2]. But, unlike the standard case, the proof requires an analysis in terms of independent blocks.

**Lemma 2.** *The following inequality holds for the expectation $\mathbf{E}_{\widetilde{S}_0}[\Phi(\widetilde{S}_0)]$ defined in terms of an independent block sequence:* $\mathbf{E}_{\widetilde{S}_0}[\Phi(\widetilde{S}_0)] \leq \mathfrak{R}_\mu^{\widetilde{D}}(H)$.

*Proof.* By the convexity of the supremum function and Jensen's inequality, $\mathbf{E}_{\widetilde{S}_0}[\Phi(\widetilde{S}_0)]$ can be bounded in terms of empirical averages over two samples:

$$\mathbf{E}_{\widetilde{S}_0}[\Phi(\widetilde{S}_0)] = \mathbf{E}_{\widetilde{S}_0}[\sup_{h\in H}\mathbf{E}_{\widetilde{S}_0'}[\widehat{R}_{\widetilde{S}_0'}(h)] - \widehat{R}_{\widetilde{S}_0}(h)] \leq \mathbf{E}_{\widetilde{S}_0,\widetilde{S}_0'}[\sup_{h\in H}\widehat{R}_{\widetilde{S}_0'}(h) - \widehat{R}_{\widetilde{S}_0}(h)].$$

We now proceed with a standard symmetrization argument with the independent blocks thought of as i.i.d. *points*:

$$\mathbf{E}_{\widetilde{S}_0}[\Phi(\widetilde{S}_0)] \leq \mathbf{E}_{\widetilde{S}_0,\widetilde{S}_0'}[\sup_{h\in H}\widehat{R}_{\widetilde{S}_0'}(h) - \widehat{R}_{\widetilde{S}_0}(h)]$$

$$= \mathbf{E}_{\widetilde{S}_0,\widetilde{S}_0'}\left[\sup_{h_a\in H_a}\frac{1}{\mu}\sum_{i=1}^{\mu}h_a(Z_i) - h_a(Z_i')\right] \qquad\qquad \text{(def. of } \widehat{R}\text{)}$$

$$= \mathbf{E}_{\widetilde{S}_0,\widetilde{S}_0',\sigma}\left[\sup_{h_a\in H_a}\frac{1}{\mu}\sum_{i=1}^{\mu}\sigma_i(h_a(Z_i) - h_a(Z_i'))\right] \qquad\qquad \text{(Rad. var.'s)}$$

$$\leq \mathbf{E}_{\widetilde{S}_0,\widetilde{S}_0',\sigma}\left[\sup_{h_a\in H_a}\frac{1}{\mu}\sum_{i=1}^{\mu}\sigma_i h_a(Z_i)\right] + \mathbf{E}_{\widetilde{S}_0,\widetilde{S}_0',\sigma}\left[\sup_{h_a\in H_a}\frac{1}{\mu}\sum_{i=1}^{\mu}\sigma_i h_a(Z_i')\right] \quad \text{(sub-add. of sup)}$$

$$= 2\mathbf{E}_{\widetilde{S}_0,\sigma}\left[\sup_{h_a\in H_a}\frac{1}{\mu}\sum_{i=1}^{\mu}\sigma_i h_a(Z_i)\right].$$

In the second equality, we introduced the Rademacher random variables $\sigma_i$s. With probability $1/2$, $\sigma_i = 1$ and the difference $h_a(Z_i) - h_a(Z_i')$ is left unchanged; and, with probability $1/2$, $\sigma_i = -1$ and $Z_i$ and $Z_i'$ are permuted. Since the blocks $Z_i$, or $Z_i'$ are independent, taking the expectation over $\sigma$ leaves the expectation unchanged. The inequality follows from the sub-additivity of the supremum function and the linearity of expectation. The final equality holds because $\widetilde{S}_0$ and $\widetilde{S}_0'$ are identically distributed due to the assumption of stationarity.

We now relate the Rademacher block sequence to a sequence over independent points. The right-hand side of the inequality just presented can be rewritten as

$$2\mathbf{E}_{\widetilde{S}_0,\sigma}\left[\sup_{h_a\in H_a}\frac{1}{\mu}\sum_{i=1}^{\mu}\sigma_i h_a(Z_i)\right] = \mathbf{E}_{\widetilde{S}_0,\sigma}\left[\sup_{h\in H}\frac{2}{\mu}\sum_{i=1}^{\mu}\sigma_i\frac{1}{a}\sum_{j=1}^{a}h(z_j^{(i)})\right],$$

where $z_j^{(i)}$ denotes the $j$th point of the $i$th block. For $j \in [1, a]$, let $\widetilde{S}_0^j$ denote the i.i.d. sample constructed from the $j$th point of each independent block $Z_i$, $i \in [1, \mu]$. By reversing the order of summations and using the convexity of the supremum function, we obtain the following:

$$
\mathop{\mathbf{E}}_{\widetilde{S}_0}[\Phi(\widetilde{S}_0)] \leq \mathop{\mathbf{E}}_{\widetilde{S}_0,\sigma} \left[ \sup_{h \in H} \frac{1}{a} \sum_{j=1}^{a} \frac{2}{\mu} \sum_{i=1}^{\mu} \sigma_i h(z_j^{(i)}) \right] \qquad \text{(reversing order of sums)}
$$

$$
\leq \frac{1}{a} \sum_{j=1}^{a} \mathop{\mathbf{E}}_{\widetilde{S}_0,\sigma} \left[ \sup_{h \in H} \frac{2}{\mu} \sum_{i=1}^{\mu} \sigma_i h(z_j^{(i)}) \right] \qquad \text{(convexity of sup)}
$$

$$
= \frac{1}{a} \sum_{j=1}^{a} \mathop{\mathbf{E}}_{\widetilde{S}_0^j,\sigma} \left[ \sup_{h \in H} \frac{2}{\mu} \sum_{i=1}^{\mu} \sigma_i h(z_j^{(i)}) \right] \qquad \text{(marginalization)}
$$

$$
= \mathop{\mathbf{E}}_{\widetilde{S}_\mu,\sigma} \left[ \sup_{h \in H} \frac{2}{\mu} \sum_{\substack{i=1 \\ z_i \in \widetilde{S}_\mu}}^{\mu} \sigma_i h(z_i) \right] \leq \mathfrak{R}_\mu^{\widetilde{D}}(H).
$$

The first equality in this derivation is obtained by marginalizing over the variables that do not appear within the inner sum. Then, the second equality holds since, by stationarity, the choice of $j$ does not change the value of the expectation. The remaining quantity, modulo absolute values, is the Rademacher complexity over $\mu$ independent points. $\qquad\square$

## 4 Non-i.i.d. Rademacher Generalization Bounds

### 4.1 General Bounds

This section presents and analyzes our main Rademacher complexity generalization bounds for stationary $\beta$-mixing sequences.

**Theorem 1** (Rademacher complexity bound). *Let $H$ be a set of hypotheses bounded by $M \geq 0$. Then, for any sample $S$ of size $m$ drawn from a stationary $\beta$-mixing distribution, and for any $\mu, a > 0$ with $2\mu a = m$ and $\delta > 2(\mu - 1)\beta(a)$, with probability at least $1 - \delta$, the following inequality holds for all hypotheses $h \in H$:*

$$
R(h) \leq \widehat{R}_S(h) + \mathfrak{R}_\mu^{\widetilde{D}}(H) + M \sqrt{\frac{\log \frac{2}{\delta'}}{2\mu}},
$$

*where $\delta' = \delta - 2(\mu - 1)\beta(a)$.*

*Proof.* Setting the right-hand side of Proposition 2 to $\delta$ and using Lemma 2 to bound $\mathbf{E}_{\widetilde{S}_0}[\Phi(\widetilde{S}_0)]$ with the Rademacher complexity $\mathfrak{R}_\mu^{\widetilde{D}}(H)$ shows the result. $\qquad\square$

As pointed out earlier, a key advantage of the Rademacher complexity is that it can be measured from data, assuming that the computation of the minimal empirical error can be done effectively and efficiently. In particular we can closely estimate $\widehat{\mathfrak{R}}_{S_\mu}(H)$, where $S_\mu$ is a subsample of the sample $S$ drawn from a $\beta$-mixing distribution, by considering random samples of $\sigma$. The following theorem gives a bound precisely with respect to the empirical Rademacher complexity $\widehat{\mathfrak{R}}_{S_\mu}$.

**Theorem 2** (Empirical Rademacher complexity bound). *Under the same assumptions as in Theorem 1, for any $\mu, a > 0$ with $2\mu a = m$ and $\delta > 4(\mu - 1)\beta(a)$, with probability at least $1 - \delta$, the following inequality holds for all hypotheses $h \in H$:*

$$
R(h) \leq \widehat{R}_S(h) + \widehat{\mathfrak{R}}_{S_\mu}(H) + 3M \sqrt{\frac{\log \frac{4}{\delta'}}{2\mu}},
$$

*where $\delta' = \delta - 4(\mu - 1)\beta(a)$.*

*Proof.* To derive this result from Theorem 1, it suffices to bound $\mathfrak{R}_\mu^{\widetilde{D}}(H)$ in terms of $\widehat{\mathfrak{R}}_{S_\mu}(H)$. The application of Corollary 1 to the indicator variable of the event $\{\mathfrak{R}_\mu^{\widetilde{D}}(H) - \widehat{\mathfrak{R}}_{S_\mu}(H) > \epsilon\}$ yields

$$\Pr\left(\mathfrak{R}_\mu^{\widetilde{D}}(H) - \widehat{\mathfrak{R}}_{S_\mu}(H) > \epsilon\right) \leq \Pr\left(\mathfrak{R}_\mu^{\widetilde{D}}(H) - \widehat{\mathfrak{R}}_{\widetilde{S}_\mu}(H) > \epsilon\right) + (\mu-1)\beta(2a-1). \quad (9)$$

Now, we can apply McDiarmid's inequality to $\mathfrak{R}_\mu^{\widetilde{D}}(H) - \widehat{\mathfrak{R}}_{\widetilde{S}_\mu}(H)$ which is defined over points drawn in an i.i.d. fashion. Changing a point of $S_\mu$ can affect $\widehat{\mathfrak{R}}_{\widetilde{S}_\mu}$ by at most $(2M/\mu)$, thus, McDiarmid's inequality gives

$$\Pr\left(\mathfrak{R}_\mu^{\widetilde{D}}(H) - \widehat{\mathfrak{R}}_{S_\mu}(H) > \epsilon\right) \leq \exp\left(\frac{-\mu\epsilon^2}{2M^2}\right) + (\mu-1)\beta(2a-1). \quad (10)$$

Note $\beta$ is a decreasing function, which implies $\beta(2a-1) \leq \beta(a)$. Thus, with probability at least $1 - \delta/2$, $\mathfrak{R}_\mu(H) \leq \widehat{\mathfrak{R}}_{S_\mu}(H) + M\sqrt{\frac{2\log\frac{1}{\delta'}}{\mu}}$, with $\delta' = \delta/2 - (\mu-1)\beta(a)$, a fortiori with $\delta' = \delta/4 - (\mu-1)\beta(a)$. The result follows this inequality combined with the statement of Theorem 1 for a confidence parameter $\delta/2$. □

This theorem can be used to derive generalization bounds for a variety of hypothesis sets and learning settings. In the next section, we present margin bounds for kernel-based classification.

## 4.2 Classification

Let $X$ denote the input space, $Y = \{-1, +1\}$ the target values in classification, and $Z = X \times Y$. For any hypothesis $h$ and margin $\rho > 0$, let $\widehat{R}_S^\rho(h)$ denote the average amount by which $yh(x)$ deviates from $\rho$ over a sample $S$: $\widehat{R}_S^\rho(h) = \frac{1}{m}\sum_{i=1}^m (\rho - y_i h(x_i))_+$. Given a positive definite symmetric kernel $K: X \times X \to \mathbb{R}$, let $\mathbf{K}$ denote its Gram matrix for the sample $S$ and $H_K$ the kernel-based hypothesis set $\{x \mapsto \sum_{i=1}^m \alpha_i K(x_i, x): \alpha \mathbf{K} \alpha^T \leq 1\}$, where $\alpha \in \mathbb{R}^{m\times 1}$ denotes the column-vector with components $\alpha_i$, $i = 1, \ldots, m$.

**Theorem 3** (Margin bound). *Let $\rho > 0$ and $K$ be a positive definite symmetric kernel. Then, for any $\mu, a > 0$ with $2\mu a = m$ and $\delta > 4(\mu-1)\beta(a)$, with probability at least $1 - \delta$ over samples $S$ of size $m$ drawn from a stationary $\beta$-mixing distribution, the following inequality holds for all hypotheses $h \in H_K$:*

$$\Pr[yh(x) \leq 0] \leq \frac{1}{\rho}\widehat{R}_S^\rho(h) + \frac{4}{\mu\rho}\sqrt{\mathrm{Tr}[\mathbf{K}]} + 3\sqrt{\frac{\log\frac{4}{\delta'}}{2\mu}},$$

*where $\delta' = \delta - 4(\mu-1)\beta(a)$.*

*Proof.* For any $h \in H$, let $\overline{h}$ denote the corresponding hypothesis defined over $Z$ by: $\forall z \in Z, \overline{h}(z) = -yh(x)$; and $\overline{H}_K$ the hypothesis set $\{z \in Z \mapsto \overline{h}(z): h \in H_K\}$. Let $L$ denote the loss function associated to the margin loss $\widehat{R}_S^\rho(h)$. Then, $\Pr[yh(x) \leq 0] \leq \Pr[(L \circ \overline{h})(z) \leq 0] = R(L \circ \overline{h})$. Since $L - 1$ is $1/\rho$-Lipschitz and $(L-1)(0) = 0$, by Talagrand's lemma [6], $\widehat{\mathfrak{R}}_S((L-1) \circ \overline{H}_K) \leq 2\widehat{\mathfrak{R}}_S(\overline{H}_K)/\rho$. The result is then obtained by applying Theorem 2 to $R((L-1) \circ \overline{h}) = R(L \circ \overline{h}) - 1$ with $\widehat{R}((L-1) \circ \overline{h}) = \widehat{R}(L \circ \overline{h}) - 1$, and using the known bound for the empirical Rademacher complexity of kernel-based classifiers [2, 11]: $\widehat{\mathfrak{R}}_S(\overline{H}_K) \leq \frac{2}{|S|}\sqrt{\mathrm{Tr}[\mathbf{K}]}$. □

In order to show that this bound converges, we must appropriately choose the parameter $\mu$, or equivalently $a$, which will depend on the mixing parameter $\beta$. In the case of algebraic mixing and using the straightforward bound $\mathrm{Tr}[\mathbf{K}] \leq mR^2$ for the kernel trace, where $R$ is the radius of the ball that contains the data, the following corollary holds.

**Corollary 2.** *With the same assumptions as in Theorem 3, if $\beta$ is further* algebraically *$\beta$-mixing, $\beta(a) = \beta_0 a^{-r}$, then, with probability at least $1 - \delta$, the following bound holds for all hypotheses $h \in H_K$:*

$$\Pr[yh(x) \leq 0] \leq \frac{1}{\rho}\widehat{R}_S^\rho(h) + \frac{8Rm^{\gamma_1}}{\rho} + 3m^{\gamma_2}\sqrt{\log\frac{4}{\delta'}},$$

*where $\gamma_1 = \frac{1}{2}\left(\frac{3}{r+2} - 1\right)$, $\gamma_2 = \frac{1}{2}\left(\frac{3}{2r+4} - 1\right)$ and $\delta' = \delta - 2\beta_0 m^{\gamma_1}$.*

This bound is obtained by choosing $\mu = \frac{1}{2}m^{\frac{2r+1}{2r+4}}$, which, modulo a multiplicative constant, is the minimizer of $(\sqrt{m}/\mu + \mu\beta(a))$. Note that for $r > 1$ we have $\gamma_1, \gamma_2 < 0$ and thus, it is clear that the bound converges, while the actual rate will depend on the distribution parameter $r$. A tighter estimate of the trace of the kernel matrix, possibly derived from data, would provide a better bound, as would stronger mixing assumptions, e.g., exponential mixing. Finally, we note that as $r \to \infty$ and $\beta_0 \to 0$, that is as the dependence between points vanishes, the right-hand side of the bound approaches $O(\widehat{R}_S^\rho + 1/\sqrt{m})$, which coincides with the asymptotic behavior in the i.i.d. case [2,4,5].

## 5   Conclusion

We presented the first Rademacher complexity error bounds for dependent samples generated by a stationary $\beta$-mixing process, a generalization of similar existing bounds derived for the i.i.d. case. We also gave the first margin bounds for kernel-based classification in this non-i.i.d. setting, including explicit bounds for algebraic $\beta$-mixing processes. Similar margin bounds can be obtained for the regression setting by using Theorem 2 and the properties of the empirical Rademacher complexity, as in the i.i.d. case. Many non-i.i.d. bounds based on other complexity measures such as the VC-dimension or covering numbers can be retrieved from our framework. Our framework and the bounds presented could serve as the basis for the design of regularization-based algorithms for dependent samples generated by a stationary $\beta$-mixing process.

### Acknowledgements

This work was partially funded by the New York State Office of Science Technology and Academic Research (NYSTAR).

## References

[1] M. Anthony and P. Bartlett. *Neural Network Learning: Theoretical Foundations*. Cambridge University Press, Cambridge, UK, 1999.

[2] P. L. Bartlett and S. Mendelson. Rademacher and Gaussian complexities: Risk bounds and structural results. *Journal of Machine Learning Research*, 3:2002, 2002.

[3] A. Irle. On the consistency in nonparametric estimation under mixing assumptions. *Journal of Multivariate Analysis*, 60:123–147, 1997.

[4] V. Koltchinskii and D. Panchenko. Rademacher processes and bounding the risk of function learning. In *High Dimensional Probability II*, pages 443–459. preprint, 2000.

[5] V. Koltchinskii and D. Panchenko. Empirical margin distributions and bounding the generalization error of combined classifiers. *Annals of Statistics*, 30, 2002.

[6] M. Ledoux and M. Talagrand. *Probability in Banach Spaces: Isoperimetry and Processes*. Springer, 1991.

[7] A. Lozano, S. Kulkarni, and R. Schapire. Convergence and consistency of regularized boosting algorithms with stationary $\beta$-mixing observations. *Advances in Neural Information Processing Systems*, 18, 2006.

[8] C. McDiarmid. On the method of bounded differences. In *Surveys in Combinatorics*, pages 148–188. Cambridge University Press, 1989.

[9] R. Meir. Nonparametric time series prediction through adaptive model selection. *Machine Learning*, 39(1):5–34, 2000.

[10] M. Mohri and A. Rostamizadeh. Stability bounds for non-iid processes. *Advances in Neural Information Processing Systems*, 2007.

[11] J. Shawe-Taylor and N. Cristianini. *Kernel Methods for Pattern Analysis*. Cambridge University Press, 2004.

[12] I. Steinwart, D. Hush, and C. Scovel. Learning from dependent observations. Technical Report LA-UR-06-3507, Los Alamos National Laboratory, 2007.

[13] L. G. Valiant. *A theory of the learnable*. ACM Press New York, NY, USA, 1984.

[14] M. Vidyasagar. *Learning and Generalization: with Applications to Neural Networks*. Springer, 2003.

[15] B. Yu. Rates of convergence for empirical processes of stationary mixing sequences. *Annals Probability*, 22(1):94–116, 1994.

